# Gaussian Processes for Regression

**Christopher K. I. Williams**
Neural Computing Research Group
Aston University
Birmingham B4 7ET, UK
c.k.i.williams@aston.ac.uk

**Carl Edward Rasmussen**
Department of Computer Science
University of Toronto
Toronto, ONT, M5S 1A4, Canada
carl@cs.toronto.edu

## Abstract

The Bayesian analysis of neural networks is difficult because a simple prior over weights implies a complex prior distribution over functions. In this paper we investigate the use of Gaussian process priors over functions, which permit the predictive Bayesian analysis for fixed values of hyperparameters to be carried out exactly using matrix operations. Two methods, using optimization and averaging (via Hybrid Monte Carlo) over hyperparameters have been tested on a number of challenging problems and have produced excellent results.

## 1 INTRODUCTION

In the Bayesian approach to neural networks a prior distribution over the weights induces a prior distribution over functions. This prior is combined with a noise model, which specifies the probability of observing the targets $t$ given function values $y$, to yield a posterior over functions which can then be used for predictions. For neural networks the prior over functions has a complex form which means that implementations must either make approximations (e.g. MacKay, 1992) or use Monte Carlo approaches to evaluating integrals (Neal, 1993).

As Neal (1995) has argued, there is no reason to believe that, for real-world problems, neural network models should be limited to nets containing only a "small" number of hidden units. He has shown that it is sensible to consider a limit where the number of hidden units in a net tends to infinity, and that good predictions can be obtained from such models using the Bayesian machinery. He has also shown that a large class of neural network models will converge to a Gaussian process prior over functions in the limit of an infinite number of hidden units.

In this paper we use Gaussian processes specified parametrically for regression problems. The advantage of the Gaussian process formulation is that the combination of

the prior and noise models can be carried out exactly using matrix operations. We also show how the *hyperparameters* which control the form of the Gaussian process can be estimated from the data, using either a maximum likelihood or Bayesian approach, and that this leads to a form of "Automatic Relevance Determination" (Mackay 1993; Neal 1995).

## 2   PREDICTION WITH GAUSSIAN PROCESSES

A stochastic process is a collection of random variables $\{Y(\boldsymbol{x})|\boldsymbol{x} \in X\}$ indexed by a set $X$. In our case $X$ will be the input space with dimension $d$, the number of inputs. The stochastic process is specified by giving the probability distribution for every finite subset of variables $Y(\boldsymbol{x}^{(1)}), \ldots, Y(\boldsymbol{x}^{(k)})$ in a consistent manner. A Gaussian process is a stochastic process which can be fully specified by its mean function $\mu(\boldsymbol{x}) = E[Y(\boldsymbol{x})]$ and its covariance function $C(\boldsymbol{x}, \boldsymbol{x}') = E[(Y(\boldsymbol{x}) - \mu(\boldsymbol{x}))(Y(\boldsymbol{x}') - \mu(\boldsymbol{x}'))]$; any finite set of points will have a joint multivariate Gaussian distribution. Below we consider Gaussian processes which have $\mu(\boldsymbol{x}) \equiv 0$.

In section 2.1 we will show how to parameterise covariances using hyperparameters; for now we consider the form of the covariance $C$ as given. The training data consists of $n$ pairs of inputs and targets $\{(\boldsymbol{x}^{(i)}, t^{(i)}), \ i = 1 \ldots n\}$. The input vector for a test case is denoted $\boldsymbol{x}$ (with no superscript). The inputs are $d$-dimensional $x_1, \ldots, x_d$ and the targets are scalar.

The predictive distribution for a test case $\boldsymbol{x}$ is obtained from the $n + 1$ dimensional joint Gaussian distribution for the outputs of the $n$ training cases and the test case, by conditioning on the observed targets in the training set. This procedure is illustrated in Figure 1, for the case where there is one training point and one test point. In general, the predictive distribution is Gaussian with mean and variance

$$
\hat{y}(\boldsymbol{x}) \;=\; \boldsymbol{k}^T(\boldsymbol{x})K^{-1}\boldsymbol{t} \tag{1}
$$

$$
\sigma_{\hat{y}}^2(\boldsymbol{x}) \;=\; C(\boldsymbol{x}, \boldsymbol{x}) - \boldsymbol{k}^T(\boldsymbol{x})K^{-1}\boldsymbol{k}(\boldsymbol{x}), \tag{2}
$$

where $\boldsymbol{k}(\boldsymbol{x}) = (C(\boldsymbol{x}, \boldsymbol{x}^{(1)}), \ldots, C(\boldsymbol{x}, \boldsymbol{x}^{(n)}))^T$, $K$ is the covariance matrix for the training cases $K_{ij} = C(\boldsymbol{x}^{(i)}, \boldsymbol{x}^{(j)})$, and $\boldsymbol{t} = (t^{(1)}, \ldots, t^{(n)})^T$.

The matrix inversion step in equations (1) and (2) implies that the algorithm has $O(n^3)$ time complexity (if standard methods of matrix inversion are employed); for a few hundred data points this is certainly feasible on workstation computers, although for larger problems some iterative methods or approximations may be needed.

### 2.1   PARAMETERIZING THE COVARIANCE FUNCTION

There are many choices of covariance functions which may be reasonable. Formally, we are required to specify functions which will generate a non-negative definite covariance matrix for any set of points $(\boldsymbol{x}^{(1)}, \ldots, \boldsymbol{x}^{(k)})$. From a modelling point of view we wish to specify covariances so that points with nearby inputs will give rise to similar predictions. We find that the following covariance function works well:

$$
C(\boldsymbol{x}^{(i)}, \boldsymbol{x}^{(j)}) \;=\; v_0 \exp\{-\frac{1}{2}\sum_{l=1}^{d} w_l(x_l^{(i)} - x_l^{(j)})^2\} \tag{3}
$$

$$
+ a_0 + a_1 \sum_{l=1}^{d} x_l^{(i)} x_l^{(j)} + v_1 \delta(i, j),
$$

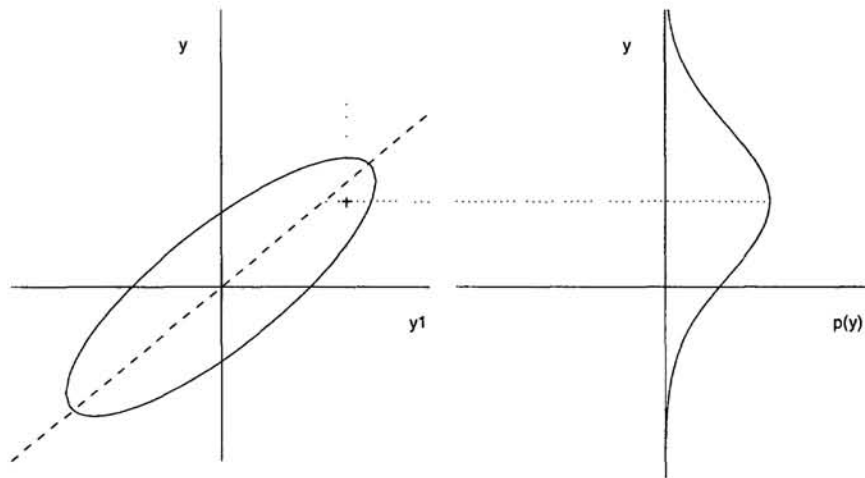

Figure 1: An illustration of prediction using a Gaussian process. There is one training case $(x^{(1)}, t^{(1)})$ and one test case for which we wish to predict $y$. The ellipse in the left-hand plot is the one standard deviation contour plot of the joint distribution of $y_1$ and $y$. The dotted line represents an observation $y_1 = t^{(1)}$. In the right-hand plot we see the distribution of the output for the test case, obtained by conditioning on the observed target. The $y$ axes have the same scale in both plots.

where $\boldsymbol{\theta} = \log(v_0, v_1, w_1, \ldots, w_d, a_0, a_1)$ plays the role of hyperparameters[1]. We define the hyperparameters to be the log of the variables in equation (4) since these are positive scale-parameters.

The covariance function is made up of three parts; the first term, a linear regression term (involving $a_0$ and $a_1$) and a noise term $v_1 \delta(i, j)$. The first term expresses the idea that cases with nearby inputs will have highly correlated outputs; the $w_l$ parameters allow a different distance measure for each input dimension. For irrelevant inputs, the corresponding $w_l$ will become small, and the model will ignore that input. This is closely related to the Automatic Relevance Determination (ARD) idea of MacKay and Neal (MacKay, 1993; Neal 1995). The $v_0$ variable gives the overall scale of the local correlations. This covariance function is valid for all input dimensionalities as compared to splines, where the integrated squared $m$th derivative is only a valid regularizer for $2m > d$ (see Wahba, 1990). $a_0$ and $a_1$ are variables controlling the scale the of bias and linear contributions to the covariance. The last term accounts for the noise on the data; $v_1$ is the variance of the noise.

Given a covariance function, the log likelihood of the training data is given by

$$l = -\frac{1}{2} \log \det K - \frac{1}{2} \boldsymbol{t}^T K^{-1} \boldsymbol{t} - \frac{n}{2} \log 2\pi. \qquad (4)$$

In section 3 we will discuss how the hyperparameters in $C$ can be adapted, in response to the training data.

## 2.2  RELATIONSHIP TO PREVIOUS WORK

The Gaussian process view provides a unifying framework for many regression methods. ARMA models used in time series analysis and spline smoothing (e.g. Wahba, 1990 and earlier references therein) correspond to Gaussian process prediction with

a particular choice of covariance function[2]. Gaussian processes have also been used in the geostatistics field (e.g. Cressie, 1993), and are known there as "kriging", but this literature has concentrated on the case where the input space is two or three dimensional, rather than considering more general input spaces.

This work is similar to Regularization Networks (Poggio and Girosi, 1990; Girosi, Jones and Poggio, 1995), except that their derivation uses a smoothness functional rather than the equivalent covariance function. Poggio *et al* suggested that the hyperparameters be set by cross-validation. The main contributions of this paper are to emphasize that a maximum likelihood solution for $\theta$ is possible, to recognize the connections to ARD and to use the Hybrid Monte Carlo method in the Bayesian treatment (see section 3).

## 3  TRAINING A GAUSSIAN PROCESS

The partial derivative of the log likelihood of the training data $l$ with respect to all the hyperparameters can be computed using matrix operations, and takes time $O(n^3)$. In this section we present two methods which can be used to adapt the hyperparameters using these derivatives.

### 3.1  MAXIMUM LIKELIHOOD

In a maximum likelihood framework, we adjust the hyperparameters so as to maximize that likelihood of the training data. We initialize the hyperparameters to random values (in a reasonable range) and then use an iterative method, for example conjugate gradient, to search for optimal values of the hyperparameters. Since there are only a small number of hyperparameters $(d+4)$ a relatively small number of iterations are usually sufficient for convergence. However, we have found that this approach is sometimes susceptible to local minima, so it is advisable to try a number of random starting positions in hyperparameter space.

### 3.2  INTEGRATION VIA HYBRID MONTE CARLO

According to the Bayesian formalism, we should start with a prior distribution $P(\theta)$ over the hyperparameters which is modified using the training data $D$ to produce a posterior distribution $P(\theta|D)$. To make predictions we then integrate over the posterior; for example, the predicted mean $\overline{y}(x)$ for test input $x$ is given by

$$\overline{y}(x) = \int \hat{y}_{\theta}(x) P(\theta|D) d\theta \tag{5}$$

where $\hat{y}_{\theta}(x)$ is the predicted mean (as given by equation 1) for a particular value of $\theta$. It is not feasible to do this integration analytically, but the Markov chain Monte Carlo method of Hybrid Monte Carlo (HMC) (Duane *et al*, 1987) seems promising for this application. We assign broad Gaussians priors to the hyperparameters, and use Hybrid Monte Carlo to give us samples from the posterior.

HMC works by creating a fictitious dynamical system in which the hyperparameters are regarded as position variables, and augmenting these with momentum variables $p$. The purpose of the dynamical system is to give the hyperparameters "inertia" so that random-walk behaviour in $\theta$-space can be avoided. The total energy, $H$, of the system is the sum of the kinetic energy, $K$, (a function of the momenta) and the potential energy, $E$. The potential energy is defined such that $p(\theta|D) \propto \exp(-E)$. We sample from the joint distribution for $\theta$ and $p$ given by $p(\theta, p) \propto \exp(-E -$

$K$); the marginal of this distribution for $\boldsymbol{\theta}$ is the required posterior. A sample of hyperparameters from the posterior can therefore be obtained by simply ignoring the momenta.

Sampling from the joint distribution is achieved by two steps: (i) finding new points in phase space with near-identical energies $H$ by simulating the dynamical system using a discretised approximation to Hamiltonian dynamics, and (ii) changing the energy $H$ by doing Gibbs sampling for the momentum variables.

Hamiltonian Dynamics

Hamilton's first order differential equations for $H$ are approximated by a discrete step (specifically using the *leapfrog* method). The derivatives of the likelihood (equation 4) enter through the derivative of the potential energy. This proposed state is then accepted or rejected using the Metropolis rule depending on the final energy $H^*$ (which is not necessarily equal to the initial energy $H$ because of the discretization). The same step size $\varepsilon$ is used for all hyperparameters, and should be as large as possible while keeping the rejection rate low.

Gibbs Sampling for Momentum Variables

The momentum variables are updated using a modified version of Gibbs sampling, thereby allowing the energy $H$ to change. A "persistence" of 0.95 is used; the new value of the momentum is a weighted sum of the previous value (with weight 0.95) and the value obtained by Gibbs sampling (weight $(1 - 0.95^2)^{1/2}$). With this form of persistence, the momenta change approximately twenty times more slowly, thus increasing the "inertia" of the hyperparameters, so as to further help in avoiding random walks. Larger values of the persistence will further increase the inertia, but reduce the rate of exploration of $H$.

Practical Details

The priors over hyperparameters are set to be Gaussian with a mean of $-3$ and a standard deviation of 3. In all our simulations a step size $\varepsilon = 0.05$ produced a very low rejection rate ($< 1\%$). The hyperparameters corresponding to $v_1$ and to the $w_l$'s were initialised to $-2$ and the rest to 0.

To apply the method we first rescale the inputs and outputs so that they have mean of zero and a variance of one on the training set. The sampling procedure is run for the desired amount of time, saving the values of the hyperparameters 200 times during the last two-thirds of the run. The first third of the run is discarded; this "burn-in" is intended to give the hyperparameters time to come close to their equilibrium distribution. The predictive distribution is then a mixture of 200 Gaussians. For a squared error loss, we use the mean of this distribution as a point estimate. The width of the predictive distribution tells us the uncertainty of the prediction.

## 4   EXPERIMENTAL RESULTS

We report the results of prediction with Gaussian process on (i) a modified version of MacKay's robot arm problem and (ii) five real-world data sets.

### 4.1   THE ROBOT ARM PROBLEM

We consider a version of MacKay's robot arm problem introduced by Neal (1995). The standard robot arm problem is concerned with the mappings

$$y_1 = r_1 \cos x_1 + r_2 \cos(x_1 + x_2) \qquad\qquad y_2 = r_1 \sin x_1 + r_2 \sin(x_1 + x_2) \qquad (6)$$

| Method | No. of inputs | sum squared test error |
|---|---|---|
| Gaussian process | 2 | 1.126 |
| Gaussian process | 6 | 1.138 |
| MacKay | 2 | 1.146 |
| Neal | 2 | 1.094 |
| Neal | 6 | 1.098 |

Table 1: Results on the robot arm task. The bottom three lines of data were obtained from Neal (1995). The MacKay result is the test error for the net with highest "evidence".

The data was generated by picking $x_1$ uniformly from [-1.932, -0.453] and [0.453, 1.932] and picking $x_2$ uniformly from [0.534, 3.142]. Neal added four further inputs, two of which were copies of $x_1$ and $x_2$ corrupted by additive Gaussian noise of standard deviation 0.02, and two further irrelevant Gaussian-noise inputs with zero mean and unit variance. Independent zero-mean Gaussian noise of variance 0.0025 was then added to the outputs $y_1$ and $y_2$. We used the same datasets as Neal and MacKay, with 200 examples in the training set and 200 in the test set.

The theory described in section 2 deals only with the prediction of a scalar quantity $Y$, so predictors were constructed for the two outputs separately, although a joint prediction is possible within the Gaussian process framework (see co-kriging, §3.2.3 in Cressie, 1993).

Two experiments were conducted, the first using only the two "true" inputs, and the second one using all six inputs. In this section we report results using maximum likelihood training; similar results were obtained with HMC. The $\log(v)$'s and $\log(w)$'s were all initialized to values chosen uniformly from [-3.0, 0.0], and were adapted separately for the prediction of $y_1$ and $y_2$ (in these early experiments the linear regression terms in the covariance function involving $a_0$ and $a_1$ were not present). The conjugate gradient search algorithm was allowed to run for 100 iterations, by which time the likelihood was changing very slowly. Results are reported for the run which gave the highest likelihood of the training data, although in fact all runs performed very similarly. The results are shown in Table 1 and are encouraging, as they indicate that the Gaussian process approach is giving very similar performance to two well-respected techniques. All of the methods obtain a level of performance which is quite close to the theoretical minimum error level of 1.0. It is interesting to look at the values of the $w$'s obtained after the optimization; for the $y_2$ task the values were 0.243, 0.237, 0.0639, $7.0 \times 10^{-4}$, $2.32 \times 10^{-6}$, $1.70 \times 10^{-6}$, and $v_0$ and $v_1$ were 7.5278 and 0.0022 respectively. The $w$ values show nicely that the first two inputs are the most important, followed by the corrupted inputs and then the irrelevant inputs. During training the irrelevant inputs are detected quite quickly, but the $w$'s for the corrupted inputs shrink more slowly, implying that the input noise has relatively little effect on the likelihood.

## 4.2   FIVE REAL-WORLD PROBLEMS

Gaussian Processes as described above were compared to several other regression algorithms on five real-world data sets in (Rasmussen, 1996; in this volume). The data sets had between 80 and 256 training examples, and the input dimension ranged from 6 to 16. The length of the HMC sampling for the Gaussian processes was from 7.5 minutes for the smallest training set size up to 1 hour for the largest ones on a R4400 machine. The results rank the methods in the order (lowest error first) a full-blown Bayesian treatment of neural networks using HMC, Gaussian

processes, ensembles of neural networks trained using cross validation and weight decay, the Evidence framework for neural networks (MacKay, 1992), and MARS. We are currently working on assessing the statistical significance of this ordering.

## 5   DISCUSSION

We have presented the method of regression with Gaussian processes, and shown that it performs well on a suite of real-world problems.

We have also conducted some experiments on the approximation of neural nets (with a finite number of hidden units) by Gaussian processes, although space limitations do not allow these to be described here. Some other directions currently under investigation include (i) the use of Gaussian processes for classification problems by softmaxing the outputs of $k$ regression surfaces (for a $k$-class classification problem), (ii) using non-stationary covariance functions, so that $C(x, x') \neq C(|x - x'|)$ and (iii) using a covariance function containing a sum of two or more terms of the form given in line 1 of equation 3.

We hope to make our code for Gaussian process prediction publically available in the near future. Check http://www.cs.utoronto.ca/neuron/delve/delve.html for details.

### Acknowledgements

We thank Radford Neal for many useful discussions, David MacKay for generously providing the robot arm data used in this paper, and Chris Bishop, Peter Dayan, Radford Neal and Huaiyu Zhu for comments on earlier drafts. CW was partially supported by EPSRC grant GR/J75425.

## Footnotes

[1] We call $\theta$ the hyperparameters as they correspond closely to hyperparameters in neural networks; in effect the weights have been integrated out exactly.

[2]Technically splines require generalized covariance functions.

## References

Cressie, N. A. C. (1993). *Statistics for Spatial Data*. Wiley.

Duane, S., Kennedy, A. D., Pendleton, B. J., and Roweth, D. (1987). Hybrid Monte Carlo. *Physics Letters B*, 195:216–222.

Girosi, F., Jones, M., and Poggio, T. (1995). Regularization Theory and Neural Networks Architectures. *Neural Computation*, 7(2):219–269.

MacKay, D. J. C. (1992). A Practical Bayesian Framework for Backpropagation Networks. *Neural Computation*, 4(3):448–472.

MacKay, D. J. C. (1993). Bayesian Methods for Backpropagation Networks. In van Hemmen, J. L., Domany, E., and Schulten, K., editors, *Models of Neural Networks II*. Springer.

Neal, R. M. (1993). Bayesian Learning via Stochastic Dynamics. In Hanson, S. J., Cowan, J. D., and Giles, C. L., editors, *Neural Information Processing Systems, Vol. 5*, pages 475–482. Morgan Kaufmann, San Mateo, CA.

Neal, R. M. (1995). *Bayesian Learning for Neural Networks*. PhD thesis, Dept. of Computer Science, University of Toronto.

Poggio, T. and Girosi, F. (1990). Networks for approximation and learning. *Proceedings of IEEE*, 78:1481–1497.

Rasmussen, C. E. (1996). A Practical Monte Carlo Implementation of Bayesian Learning. In Touretzky, D. S., Mozer, M. C., and Hasselmo, M. E., editors, *Advances in Neural Information Processing Systems 8*. MIT Press.

Wahba, G. (1990). *Spline Models for Observational Data*. Society for Industrial and Applied Mathematics. CBMS-NSF Regional Conference series in applied mathematics.
